# Maximum entropy discrimination

**Tommi Jaakkola**
MIT AI Lab
545 Technology Sq.
Cambridge, MA 02139
*tommi@ai.mit.edu*

**Marina Meila**
MIT AI Lab
545 Technology Sq.
Cambridge, MA 02139
*mmp@ai.mit.edu*

**Tony Jebara**
MIT Media Lab
20 Ames St.
Cambridge, MA 02139
*jebara@media.mit.edu*

## Abstract

We present a general framework for discriminative estimation based on the maximum entropy principle and its extensions. All calculations involve distributions over structures and/or parameters rather than specific settings and reduce to relative entropy projections. This holds even when the data is not separable within the chosen parametric class, in the context of anomaly detection rather than classification, or when the labels in the training set are uncertain or incomplete. Support vector machines are naturally subsumed under this class and we provide several extensions. We are also able to estimate exactly and efficiently discriminative distributions over tree structures of class-conditional models within this framework. Preliminary experimental results are indicative of the potential in these techniques.

## 1 Introduction

Effective discrimination is essential in many application areas. Employing generative probability models such as mixture models in this context is attractive but the criterion (e.g., maximum likelihood) used for parameter/structure estimation is suboptimal. Support vector machines (SVMs) are, for example, more robust techniques as they are specifically designed for discrimination [9].

Our approach towards general discriminative training is based on the well known maximum entropy principle (e.g., [3]). This enables an appropriate training of both ordinary and structural parameters of the model (cf. [5, 7]). The approach is not limited to probability models and extends, e.g., SVMs.

## 2 Maximum entropy classification

Consider a two-class classification problem[1] where labels $y \in \{-1, 1\}$ are assigned

to examples $X \in \mathcal{X}$. Given two generative probability distributions $P(X|\theta_y)$ with parameters $\theta_y$, one for each class, the corresponding decision rule follows the sign of the *discriminant function*:

$$\mathcal{L}(X|\Theta) = \log \frac{P(X|\theta_1)}{P(X|\theta_{-1})} + b \tag{1}$$

where $\Theta = \{\theta_1, \theta_{-1}, b\}$ and $b$ is a bias term, usually expressed as a log-ratio $b = \log p/(1-p)$. The class-conditional distributions may come from different families of distributions or the parametric discriminant function could be specified directly without any reference to models. The parameters $\theta_y$ may also include the model structure (see later sections).

The parameters $\Theta = \{\theta_1, \theta_{-1}, b\}$ should be chosen to maximize classification accuracy. We consider here the more general problem of finding a distribution $P(\Theta)$ over parameters and using a convex combination of discriminant functions, i.e., $\int P(\Theta)\mathcal{L}(X|\Theta)d\Theta$ in the decision rule. The search for the optimal $P(\Theta)$ can be formalized as a *maximum entropy* (ME) estimation problem. Given a set of training examples $\{X_1, \ldots, X_T\}$ and corresponding labels $\{y_1, \ldots, y_T\}$ we find a distribution $P(\Theta)$ that maximizes the entropy $H(P)$ subject to the classification constraints $\int P(\Theta) \left[ y_t \mathcal{L}(X_t|\Theta) \right] d\Theta \geq \gamma$ for all $t$. Here $\gamma > 0$ specifies a desired classification margin. The solution is unique (if it exists) since $H(P)$ is concave and the linear constraints specify a convex region. Note that the preference towards high entropy distributions (fewer assumptions) applies only within the admissible set of distributions $\mathcal{P}_\gamma$ consistent with the constraints. See [2] for related work.

We will extend this basic idea in a number of ways. The ME formulation assumes, for example, that the training examples can be separated with the specified margin. We may also have a reason to prefer some parameter values over others and would therefore like to incorporate a prior distribution $P_0(\Theta)$. Other extensions and generalizations will be discussed later in the paper.

A more complete formulation is based on the following *minimum relative entropy* principle:

**Definition 1** *Let $\{X_t, y_t\}$ be the training examples and labels, $\mathcal{L}(X|\Theta)$ a parametric discriminant function, and $\gamma = [\gamma_1, \ldots, \gamma_t]$ a set of margin variables. Assuming a prior distribution $P_0(\Theta, \gamma)$, we find the discriminative minimum relative entropy (MRE) distribution $P(\Theta, \gamma)$ by minimizing $D(P\|P_0)$ subject to*

$$\int P(\Theta, \gamma) \left[ y_t \mathcal{L}(X_t|\Theta) - \gamma_t \right] d\Theta d\gamma \geq 0 \tag{2}$$

*for all $t$. Here $\hat{y} = \text{sign}\left( \int P(\Theta)\mathcal{L}(X|\Theta)d\Theta \right)$ specifies the decision rule for any new example $X$.*

The margin constraints and the preference towards large margin solutions are encoded in the prior $P_0(\gamma)$. Allowing negative margin values with non-zero probabilities also guarantees that the admissible set $\mathcal{P}$ consisting of distributions $P(\Theta, \gamma)$ consistent with the constraints, is never empty. Even when the examples cannot be separated by any discriminant function in the parametric class (e.g., linear), we get a valid solution. The miss-classification penalties follow from $P_0(\gamma)$ as well.

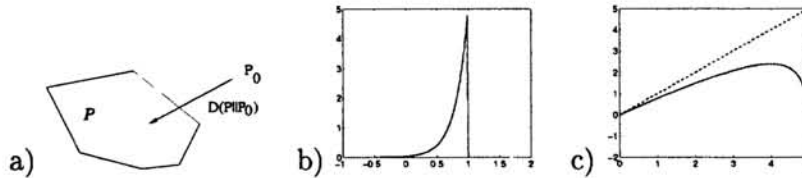

Figure 1: a) Minimum relative entropy (MRE) projection from the prior distribution to the admissible set. b) The margin prior $P_0(\gamma_t)$. c) The potential terms in the MRE formulation (solid line) and in SVMs (dashed line). $c = 5$ in this case.

Suppose $P_0(\Theta, \gamma) = P_0(\Theta) P_0(\gamma)$ and $P_0(\gamma) = \prod_t P_0(\gamma_t)$, where

$$P_0(\gamma_t) = c\, e^{-c(1-\gamma_t)} \quad \text{for } \gamma_t \leq 1, \tag{3}$$

This is shown in Figure 1b. The penalty for margins smaller than $1 - 1/c$ (the prior mean of $\gamma_t$) is given by the relative entropy distance between $P(\gamma)$ and $P_0(\gamma)$. This is similar but not identical to the use of slack variables in support vector machines. Other choices of the prior are discussed in [4].

The MRE solution can be viewed as a relative entropy projection from the prior distribution $P_0(\Theta, \gamma)$ to the admissible set $\mathcal{P}$. Figure 1a illustrates this view. From the point of view of regularization theory, the prior probability $P_0$ specifies the entropic regularization used in this approach.

**Theorem 1** *The solution to the MRE problem has the following general form [1]*

$$P(\Theta, \gamma) = \frac{1}{Z(\lambda)} P_0(\Theta, \gamma)\, e^{\sum_t \lambda_t [y_t \mathcal{L}(X_t | \Theta) - \gamma_t]} \tag{4}$$

*where $Z(\lambda)$ is the normalization constant (partition function) and $\lambda = \{\lambda_1, \ldots, \lambda_T\}$ defines a set of non-negative Lagrange multipliers, one for each classification constraint. $\lambda$ are set by finding the unique maximum of the following jointly concave objective function: $J(\lambda) = -\log Z(\lambda)$*

The solution is *sparse*, i.e., only a few Lagrange multipliers will be non-zero. This arises because many of the classification constraints become irrelevant once the constraints are enforced for a small subset of examples. Sparsity leads to immediate but weak generalization guarantees expressed in terms of the number of non-zero Lagrange multipliers [4]. Practical leave-one-out cross-validation estimates can be also derived.

## 2.1   Practical realization of the MRE solution

We now turn to finding the MRE solution. To begin with, we note that any disjoint factorization of the prior $P_0(\Theta, \gamma)$, where the corresponding parameters appear in distinct additive components in $y_t \mathcal{L}(X_t, \Theta) - \gamma_t$, leads to a disjoint factorization of the MRE solution $P(\Theta, \gamma)$. For example, $\{\Theta \setminus b, b, \gamma\}$ provides such a factorization. As a result of this factorization, the bias term could be eliminated by imposing additional constraints on the Lagrange multipliers [4]. This is analogous to the handling of the bias term in support vector machines [9].

We consider now a few specific realizations such as support vector machines and a class of graphical models.

### 2.1.1 Support vector machines

It is well known that the log-likelihood ratio of two Gaussian distributions with equal covariance matrices yields a linear decision rule. With a few additional assumptions, the MRE formulation gives support vector machines:

**Theorem 2** *Assuming $\mathcal{L}(X, \Theta) = \theta^T X - b$ and $P_0(\Theta, \gamma) = P_0(\theta) P_0(b) P_0(\gamma)$ where $P_0(\theta)$ is $N(0, I)$, $P_0(b)$ approaches a non-informative prior, and $P_0(\gamma)$ is given by eq. (3) then the Lagrange multipliers $\lambda$ are obtained by maximizing $J(\lambda)$ subject to $0 \leq \lambda_t \leq c$ and $\sum_t \lambda_t y_t = 0$, where*

$$J(\lambda) = \sum_t [\lambda_t + \log(1 - \lambda_t/c)] - \frac{1}{2} \sum_{t,t'} \lambda_t \lambda_{t'} y_t y_{t'} (X_t^T X_{t'}) \qquad (5)$$

The only difference between our $J(\lambda)$ and the (dual) optimization problem for SVMs is the additional potential term $\log(1 - \lambda_t/c)$. This highlights the effect of the different miss-classification penalties, which in our case come from the MRE projection. Figure 1b shows, however, that the additional potential term does not always carry a huge effect (for $c = 5$). Moreover, in the separable case, letting $c \to \infty$, the two methods coincide. The decision rules are formally identical.

We now consider the case where the discriminant function $\mathcal{L}(X, \Theta)$ corresponds to the log-likelihood ratio of two Gaussians with different (and adjustable) covariance matrices. The parameters $\Theta$ in this case are both the means and the covariances. The prior $P_0(\Theta)$ must be the conjugate Normal-Wishart to obtain closed form integrals[2] for the partition function, $Z$. Here, $P(\Theta_1, \Theta_{-1})$ is $P(m_1, V_1) P(m_{-1}, V_{-1})$, a density over means and covariances.

The prior distribution has the form $P_0(\Theta_1) = \mathcal{N}(m_1; m_0, V_1/k) \, \mathcal{IW}(V_1; kV_0, k)$ with parameters $(k, m_0, V_0)$ that can be specified manually or one may let $k \to 0$ to get a non-informative prior. Integrating over the parameters and the margin, we get $Z = Z_\gamma \times Z_1 \times Z_{-1}$, where

$$Z_1 \propto N_1^{-d/2} |\pi S_1|^{-N_1/2} \, \Pi_{j=1}^d \Gamma((N_1 + 1 - j)/2) \qquad (6)$$

$N_1 \triangleq \sum_t w_t$, $\bar{X}_1 \triangleq \sum_t \frac{w_t}{N_1} X_t$, $S_1 \triangleq \sum_t w_t X_t X_t^T - N_1 \bar{X}_1 \bar{X}_1^T$. Here, $w_t$ is a scalar weight given by $w_t = u(y_t) + y_t \lambda_t$. For $Z_{-1}$, the weights are set to $w_t = u(-y_t) - y_t \lambda_t$; $u(\cdot)$ is the step function. Given $Z$, updating $\lambda$ is done by maximizing $J(\lambda)$. The resulting marginal MRE distribution over the parameters (normalized by $Z_1 \times Z_{-1}$) is a Normal-Wishart distribution itself, $P(\Theta_1) = \mathcal{N}(m_1; \bar{X}_1, V_1/N_1) \, \mathcal{IW}(V_1; S_1, N_1)$ with the final $\lambda$ values. Predicting the label for a new example $X$ involves taking expectations of the discriminant function under a Normal-Wishart. This is

$$E_{P(\Theta_1)}[\log P(X|\Theta_1)] = \text{constant} - \frac{N_1}{2}(X - \bar{X}_1)^T S_1^{-1}(X - \bar{X}_1) \qquad (7)$$

We thus obtain discriminative *quadratic* decision boundaries. These extend the linear boundaries without (explicitly) resorting to *kernels*. More generally, the covariance estimation in this framework adaptively modifies the kernel.

### 2.1.2 Graphical models

We consider here graphical models with no hidden variables. The ME (or MRE) distribution is in this case a distribution over both structures and parameters. Finding the distribution over parameters can be done in closed form for conjugate priors when the observations are complete. The distribution over structures is, in general, intractable. A notable exception is a tree model that we discuss in the forthcoming.

A tree graphical model is a graphical model for which the structure is a *tree*. This model has the property that its log-likelihood can be expressed as a sum of local terms [8]

$$\log P(X, E|\theta) = \sum_u h_u(X, \theta) + \sum_{uv \in E} w_{uv}(X, \theta) \tag{8}$$

The discriminant function consisting of the log-likelihood ratio of a pair of tree models (depending on the edge sets $E_1$, $E_{-1}$, and parameters $\theta_1$, $\theta_{-1}$) can be also expressed in this form.

We consider here the ME distribution over tree structures for fixed parameters[3]. The treatment of the general case (i.e. including the parameters) is a direct extension of this result. The ME distribution over the edge sets $E_1$ and $E_{-1}$ factorizes with components

$$P(E_{\pm 1}) = \frac{1}{Z_{\pm 1}} e^{\pm \sum_t \lambda_t y_t \left[ \sum_{uv \in E_{\pm 1}} w_{uv}^{\pm 1}(X_t, \theta_{\pm 1}) + \sum_u h_u(X_t, \theta_{\pm 1}) \right]} = \frac{h^{\pm 1}}{Z_{\pm 1}} \prod_{uv \in E_{\pm 1}} W_{uv}^{\pm 1} \tag{9}$$

where $Z_{\pm 1}, h^{\pm 1}, W^{\pm 1}$ are functions of the same Lagrange multipliers $\lambda$. To completely define the distribution we need to find $\lambda$ that optimize $J(\lambda)$ in Theorem 1; for classification we also need to compute averages with respect to $P(E_{\pm 1})$. For these, it suffices to obtain an expression of the partition function(s) $Z_{\pm 1}$.

$P$ is a discrete distribution over all possible tree structures for $n$ variables (there are $n^{n-2}$ trees). However, a remarkable graph theory result, called the *Matrix Tree Theorem* [10], enables us to perform all necessary summations in closed form in polynomial time. On the basis of this result, we find

**Theorem 3** *The normalization constant $Z$ of a distribution of the form (9) is*

$$Z = h \cdot \sum_E \prod_{uv \in E} W_{uv} = h \cdot |Q(W)|, \quad \text{where} \tag{10}$$

$$Q_{uv}(W) = \begin{cases} -W_{uv} & u \neq v \\ \sum_{v'=1}^n W_{v'v} & u = v \end{cases} \tag{11}$$

This shows that summing over the distribution of all trees, when this distribution factors according to the trees' edges, can be done in closed form by computing the value of a determinant in time $\mathcal{O}(n^3)$. Since we obtain a closed form expression, optimization of the Lagrange multipliers and evaluating the resulting classification rule are also tractable.

Figure 2a provides a comparison of the discriminative tree approach and a maximum likelihood tree estimation method on a DNA splice junction problem.

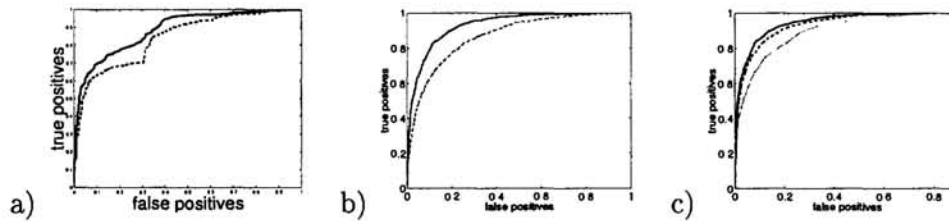

Figure 2: ROC curves based on independent test sets. a) Tree estimation: discriminative (solid) and ML (dashed) trees. b) Anomaly detection: MRE (solid) and Bayes (dashed). c) Partially labeled case: 100% labeled (solid), 10% labeled + 90% unlabeled (dashed), and 10% labeled + 0% unlabeled training examples (dotted).

## 3   Extensions

**Anomaly detection:** In anomaly detection we are given a set of training examples representing only one class, the "typical" examples. We attempt to capture regularities among the examples to be able to recognize unlikely members of this class. Estimating a probability distribution $P(X|\theta)$ on the basis of the training set $\{X_1, \ldots, X_T\}$ via the ML (or analogous) criterion is not appropriate; there is no reason to further increase the probability of those examples that are already well captured by the model. A more relevant measure involves the level sets $\mathcal{X}_\gamma = \{X \in \mathcal{X} : \log P(X|\theta) \geq \gamma\}$ which are used in deciding the class membership in any case. We estimate the parameters $\theta$ to optimize an appropriate level set.

**Definition 2** *Given a probability model $P(X|\theta)$, $\theta \in \Theta$, a set of training examples $\{X_1, \ldots, X_T\}$, a set of margin variables $\gamma = [\gamma_1, \ldots, \gamma_T]$, and a prior distribution $P_0(\theta, \gamma)$ we find the MRE distribution $P(\theta, \gamma)$ such that minimizes $D(P\|P_0)$ subject to the constraints $\int P(\theta, \gamma)\,[\log P(X_t|\theta) - \gamma_t]\,d\theta d\gamma \geq 0$ for all $t = 1, \ldots, T$.*

Note that this again a MRE projection whose solution can be obtained as before. The choice of $P_0(\gamma)$ in $P_0(\theta, \gamma) = P_0(\theta)P_0(\gamma)$ is not as straightforward as before since each margin $\gamma_t$ needs to be close to achievable log-probabilities. We can nevertheless find a reasonable choice by relating the prior mean of $\gamma_t$ to some $\alpha-$percentile of the training set log-probabilities generated through ML or other estimation criterion. Denote the resulting value by $l_\alpha$ and define the prior $P_0(\gamma_t)$ as $P_0(\gamma_t) = c\,e^{-c\,(l_\alpha - \gamma_t)}$ for $\gamma_t \leq l_\alpha$. In this case the prior mean of $\gamma_t$ is $l_\alpha - 1/c$.

Figure 2b shows in the context of a simple product distribution that this choice of prior together with the MRE framework leads to a real improvement over standard (Bayesian) approach. We believe, however, that the effect will be more striking for sophisticated models such as HMMs that may otherwise easily capture spurious regularities in the data. An extension of this formalism to latent variable models is provided in [4].

**Uncertain or incompletely labeled examples:** Examples with uncertain labels are hard to deal with in any (probabilistic or not) discriminative classification method. Uncertain labels can be, however, handled within the maximum entropy formalism: let $y = \{y_1, \ldots, y_T\}$ be a set of binary variables corresponding to the labels for the training examples. We can define a prior uncertainty over the labels by specifying $P_0(y)$; for simplicity, we can take this to be a product distribution

$P_0(y) = \prod_t P_{t,0}(y_t)$ where a different level of uncertainty can be assigned to each example. Consequently, we find the minimum relative entropy projection from the prior distribution $P_0(\Theta, \gamma, y) = P_0(\Theta) P_0(\gamma) P_0(y)$ to the admissible set of distributions (no longer a function of the labels) that are consistent with the constraints: $\sum_y \int_{\Theta, \gamma} P(\Theta, \gamma, y) [y_t \mathcal{L}(X_t, \Theta) - \gamma_t] \, d\Theta \, d\gamma \geq 0$ for all $t = 1, \ldots, T$. The MRE principle differs from *transduction* [9], provides a soft rather than hard assignment of unlabeled examples, and is fundamentally driven by large margin classification. The MRE solution is not, however, often feasible to obtain in practice. We can nevertheless formulate an efficient mean field approach in this context [4]. Figure 2c demonstrates that even the approximate method is able to reap most of the benefit from unlabeled examples (compare, e.g., [6]). The results are for a DNA splice junction classification problem. For more details see [4].

## 4  Discussion

We have presented a general approach to discriminative training of model parameters, structures, or parametric discriminant functions. The formalism is based on the minimum relative entropy principle reducing all calculations to relative entropy projections. The idea naturally extends beyond standard classification and covers anomaly detection, classification with partially labeled examples, and feature selection.

## Footnotes

[1]The extension to a multi-class is straightforward[4]. The formulation also admits an easy extension to regression problems, analogously to SVMs.

[2]This can be done more generally for conjugate priors in the exponential family.

[3]Each tree relies on a different set of $n-1$ pairwise node marginals. In our experiments the class-conditional pairwise marginals were obtained directly from data.

## References

[1] Cover and Thomas (1991). *Elements of information theory*. John Wiley & Sons.

[2] Kivinen J. and Warmuth M. (1999). Boosting as Entropy Projection. *Proceedings of the 12th Annual Conference on Computational Learning Theory*.

[3] Levin and Tribus (eds.) (1978). *The maximum entropy formalism*. Proceedings of the Maximum entropy formalism conference, MIT.

[4] Jaakkola T., Meilă M. and Jebara T. (1999). Maximum entropy discrimination. MIT AITR-1668, `http://www.ai.mit.edu/~tommi/papers.html`.

[5] Jaakkola T. and Haussler D. (1998). Exploiting generative models in discriminative classifiers. NIPS 11.

[6] Joachims, T. (1999). Transductive inference for text classification using support vector machines. *International conference on Machine Learning*.

[7] Jebara T. and Pentland A. (1998). Maximum conditional likelihood via bound maximization and the CEM algorithm. NIPS 11.

[8] Meilă M. and Jordan M. (1998). Estimating dependency structure as a hidden variable. NIPS 11.

[9] Vapnik V. (1998). *Statistical learning theory*. John Wiley & Sons.

[10] West D. (1996). *Introduction to graph theory*. Prentice Hall.
